# Target Neighbor Consistent Feature Weighting for Nearest Neighbor Classification

**Ichiro Takeuchi**
Department of Engineering
Nagoya Institute of Technology
takeuchi.ichiro@nitech.ac.jp

**Masashi Sugiyama**
Department of Computer Science
Tokyo Institute of Technology
sugi@cs.titech.ac.jp

## Abstract

We consider feature selection and weighting for nearest neighbor classifiers. A technical challenge in this scenario is how to cope with discrete update of nearest neighbors when the feature space metric is changed during the learning process. This issue, called the *target neighbor change*, was not properly addressed in the existing feature weighting and metric learning literature. In this paper, we propose a novel feature weighting algorithm that can *exactly* and efficiently keep track of the correct target neighbors via sequential quadratic programming. To the best of our knowledge, this is the first algorithm that guarantees the consistency between target neighbors and the feature space metric. We further show that the proposed algorithm can be naturally combined with *regularization path tracking*, allowing computationally efficient selection of the regularization parameter. We demonstrate the effectiveness of the proposed algorithm through experiments.

## 1 Introduction

*Nearest neighbor (NN) classifiers* would be one of the classical and perhaps the simplest non-linear classification algorithms. Nevertheless, they have gathered considerable attention again recently since they are demonstrated to be highly useful in state-of-the-art real-world applications [1, 2]. For further enhancing the accuracy and interpretability of NN classifiers, feature extraction and feature selection are highly important. Feature extraction for NN classifiers has been addressed by the name of *metric learning* [3–6], while feature selection for NN classifiers has been studied by the name of *feature weighting* [7–11].

One of the fundamental approaches to feature extraction/selection for NN classifiers is to learn the feature metric/weights so that instance pairs in the same class ('*must-link*') are close and instance pairs in other classes ('*cannot-link*') are far apart [12, 13]. Although this approach tends to provide simple algorithms, it does not have direct connection to the classification *loss* for NN classifiers, and thus its validity is not clear.

However, directly incorporating the NN classification loss involves a significant technical challenge called the *target neighbor (TN) change*. To explain this, let us consider binary classification by a 3NN classifier (see Figure 1). Since the classification result is determined by the majority vote from 3 nearest instances, the classification loss is defined using the distance to the 2nd nearest instance in each class (which is referred to as a TN; see Section 2 for details). However, since '*nearest*' instances are generally changed when feature metric/weights are updated, TNs must also be updated to be kept *consistent* with the learned feature metric/weights during the learning process.

Although the TN change is a fundamental requirement in feature extraction/selection for NN classifiers, existing methods did not handle this issue properly. For example, in a seminal feature weighting method called *Relief* [7, 8], the fixed TNs determined based on the uniform weights (i.e., the Euclidean distance) are used throughout the learning process. Thus, the TN-weight consistency is

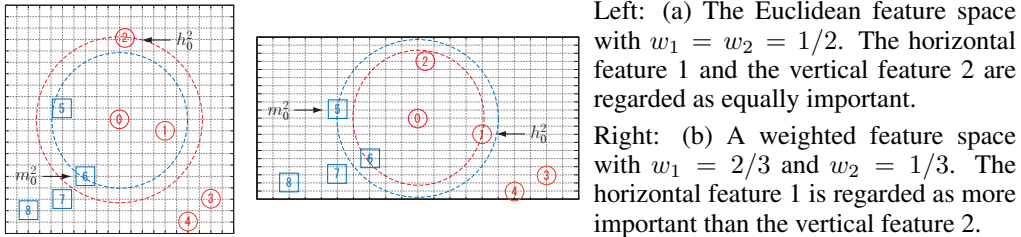

Left: (a) The Euclidean feature space with $w_1 = w_2 = 1/2$. The horizontal feature 1 and the vertical feature 2 are regarded as equally important.

Right: (b) A weighted feature space with $w_1 = 2/3$ and $w_2 = 1/3$. The horizontal feature 1 is regarded as more important than the vertical feature 2.

Figure 1: Illustration of target neighbors (TNs). An instance ⓪ in the middle is correctly classified in 3NN classification if the distance to the 2nd nearest instance in the same class (called 2nd *target hit* and denoted by $h_0^2$) is smaller than the distance to the 2nd nearest instance in different classes (called 2nd *target miss* and denoted by $m_0^2$). In the Euclidean feature space (a), the 2nd target hit/miss are given by $(h_0^2, m_0^2) = (②, 6)$. Since $d(x_0, x_2|w) > d(x_0, x_6|w)$, the instance ⓪ is misclassified. On the other hand, in the weighted feature space (b), the 2nd target hit/miss are given by $(h_0^2, m_0^2) = (①, 5)$. Since $d(x_0, x_1|w) < d(x_0, x_5|w)$, the instance ⓪ is correctly classified.

not guaranteed (*large-margin metric learning* [5] also suffers from the same drawback). The *Simba* algorithm [9] is a maximum-margin feature weighting method which adaptively updates TNs in the online learning process. However, the TN-weight consistency is not still guaranteed in Simba. *I-Relief* [10, 11] is a feature weighting method which cleverly avoids the TN change problem by considering a stochastic variant of NN classifiers (*neighborhood component analysis* [4] also introduced similar stochastic approximation). However, since the behavior of stochastic NN classifiers tends to be significantly different from the original ones, the obtained feature metric/weights are not necessarily useful for the original NN classifiers.

In this paper, we focus on the feature selection (i.e., feature weighting) scenario, and propose a novel method that can properly address the TN change problem. More specifically, we formulate feature weighting as a regularized empirical risk minimization problem, and develop an algorithm that *exactly* and efficiently keeps track of the correct TNs via *sequential quadratic programming*. To the best of our knowledge, this is the first algorithm that systematically handles TN-changes and guarantees the TN-weight consistency. We further show that the proposed algorithm can be naturally combined with *regularization path tracking* [14], allowing computationally efficient selection of the regularization parameter. Finally, we demonstrate the effectiveness of the proposed algorithm through experiments.

Throughout the paper, the superscript $\top$ indicates the transpose of vectors or matrices. We use $\mathbb{R}$ and $\mathbb{R}_+$ to denote the sets of real numbers and non-negative real numbers, respectively, while we use $\mathbb{N}_n := \{1, \ldots, n\}$ to denote the set of natural numbers. The notations $\mathbf{0}$ and $\mathbf{1}$ indicate vectors or matrices with all 0 and 1, respectively. The number of elements in a set $\mathcal{S}$ is denoted by $|\mathcal{S}|$.

## 2 Preliminaries

In this section, we formulate the problem of feature weighting for nearest neighbor (NN) classification, and explain the fundamental concept of *target neighbor (TN) change*.

Consider a classification problem from $n$ training instances with $\ell$ features. Let $x_i := [x_{i1} \ldots x_{i\ell}]^\top \in \mathbb{R}^\ell$ be the $i$-th training instance and $y_i$ be the corresponding label. The squared Euclidean distance between two instances $x_i$ and $x_{i'}$ is $\sum_{j \in \mathbb{N}_\ell} (x_{ij} - x_{i'j})^2$, while the weighted squared Euclidean distance is written as

$$d(x_i, x_{i'}|w) := \sum_{j \in \mathbb{N}_\ell} w_j(x_{ij} - x_{i'j})^2 = \varepsilon_{i,i'}^\top w, \tag{1}$$

where $w := [w_1 \ldots w_\ell] \in [0, 1]^\ell$ is an $\ell$-dimensional vector of non-negative weights and $\varepsilon_{i,i'} := [(x_{i1} - x_{i'1})^2 \ldots (x_{i\ell} - x_{i'\ell})^2]^\top \in \mathbb{R}^\ell$, $(i, i') \in \mathbb{N}_n \times \mathbb{N}_n$, is introduced for notational simplicity.

We develop a feature weighting algorithm within the framework of regularized empirical risk minimization, i.e., minimizing the linear combination of a loss term and a regularization term. In order to formulate the loss term for NN classification, let us introduce the notion of *target neighbors (TNs)*:

**Definition 1 (Target neighbors (TNs))** *Define* $\mathbb{H}_i := \{h \in \mathbb{N}_n | y_h = y_i, h \neq i\}$ *and* $\mathbb{M}_i := \{m \in \mathbb{N}_n | y_m \neq y_i\}$ *for* $i \in \mathbb{N}_n$. *Given a weight vector $w$, an instance $h \in \mathbb{H}_i$ is said to be* the $\kappa$-th target hit *of an instance $i$ if it is the $\kappa$-th nearest instance among $\mathbb{H}_i$, and $m \in \mathbb{M}_i$ is said to be* the $\lambda$-th target miss *of an instance $i$ if it is the $\lambda$-th nearest instance among $\mathbb{M}_i$, where the distance between instances are measured by the weighted Euclidean distance (1). The $\kappa$-th target hit and $\lambda$-th target miss of an instance $i \in \mathbb{N}_n$ are denoted by $h_i^\kappa$ and $m_i^\lambda$, respectively. Target hits and misses are collectively called as target neighbors (TNs)* [1].

Using TNs, the misclassification rate of a binary $k$NN classifier when $k$ is odd is formulated as $L^{k\text{NN}}(w) := n^{-1} \sum_{i \in \mathbb{N}_n} I\{d(x_i, x_{h_i^\kappa}|w) > d(x_i, x_{m_i^\lambda}|w)\}$ with $\kappa = \lambda = (k+1)/2$, where $I(\cdot)$ is the indicator function with $I(z) = 1$ if $z$ is true and $I(z) = 0$ otherwise. For example, in binary 3NN classification, an instance is misclassified if and only if the distance to the 2nd target hit is larger than the distance to the 2nd target miss (see Figure 1). The misclassification cost of a multi-class problem can also be formulated by using TNs similarly, but we omit the details for the sake of simplicity.

Since the indicator function $I(\cdot)$ included in the loss function $L^{k\text{NN}}(w)$ is hard to directly deal with, we introduce the *nearest neighbor (NN) margin* [2] as a surrogate:

**Definition 2 (Nearest neighbor (NN) margin)** *Given a weight vector $w$, the $(\kappa, \lambda)$-neighbor margin is defined as $d(x_i, x_{m_i^\lambda}|w) - d(x_i, x_{h_i^\kappa}|w)$ for $i \in \mathbb{N}_n$, $\kappa \in \mathbb{N}_{|\mathbb{H}_i|}$, and $\lambda \in \mathbb{N}_{|\mathbb{M}_i|}$.*

Based on the NN margin, our loss function is defined as $L(w) := n^{-1} \sum_{i \in \mathbb{N}_n} \big(d(x_i, x_{h_i^\kappa}|w) - d(x_i, x_{m_i^\lambda}|w)\big)$. By minimizing $L(w)$, the average $(\kappa, \lambda)$-neighbor margin over all instances is maximized. This loss function allows us to find feature weights such that the distance to the $\kappa$-th target hit is as small as possible, while the distance to the $\lambda$-th target miss is as large as possible.

A regularization term is introduced for incorporating our prior knowledge on the weight vector. Let $\bar{w} \in [0,1]^\ell$ be our prior weight vector, and we use the regularization term of the form $\Omega(w) := \frac{1}{2}||w - \bar{w}||_2^2$. For example, if we choose $\bar{w} := \ell^{-1}\mathbf{1}$, it implies that our baseline choice of the feature weights is uniform, i.e., the Euclidean distance metric [6].

Given the loss term $L(w)$ and the regularization term $\Omega(w)$, the feature weighting problem we are going to study in this paper is formulated as

$$\min_w \quad \theta n^{-1} \sum_{i \in \mathbb{N}_n} \big(d(x_i, x_{h_i^\kappa}|w) - d(x_i, x_{m_i^\lambda}|w)\big) + \frac{1}{2}||w - \bar{w}||_2^2 \quad \text{s.t. } \mathbf{1}^\top w = 1, \ w \geq \mathbf{0}, \quad (2)$$

where $\theta \in \mathbb{R}_+$ is a regularization parameter for controlling the balance between the loss term $L(w)$ and the regularization term $\Omega(w)$. The first equality constraint restricts that the sum of the weights to be one, while the second constraint indicates that the weights are non-negative. The former is introduced for fixing the scale of the distance metric.

It is important to note that TNs $\{(h_i^\kappa, m_i^\lambda)\}_{i \in \mathbb{N}_n}$ are dependent on the weights $w$ because the weighted Euclidean distance (1) is used in their definitions. Thus, we need to properly update TNs in the optimization process. We refer to this problem as the *target neighbor change* (TN-change) problem. Since TNs change in a discrete fashion with respect to the weights $w$, the problem (2) has a non-smooth and non-convex objective function. In the next section, we introduce an algorithm for finding a local minimum solution of (2). An advantage of the proposed algorithm is that it monotonically decreases the objective function in (2), while TNs are properly updated so that they are always kept *consistent* with the feature space metric given by the weights $w$ in the following sense:

**Definition 3 (TN-weight Consistency)** *A weight vector $w$ and $n$ pairs of instances $\{(h_i^\kappa, m_i^\lambda)\}_{i \in \mathbb{N}_n}$ are said to be TN-weight consistent if $\{(h_i^\kappa, m_i^\lambda)\}_{i \in \mathbb{N}_n}$ are the TNs when the distance is measured by the weighted Euclidean distance (1) using the weights $w$.*

Figure 1 illustrates how TNs are defined. In the Euclidean feature space with $w_1 = w_2 = 1/2$, the 2nd target hit and miss of the instance ⓪ are given by $(h_0^2, m_0^2) = (②, \boxed{6})$. Since $d(x_0, x_2|w) > d(x_0, x_6|w)$, the instance ⓪ is misclassified in 3NN classification. On the other hand, in the weighted feature space with $(w_1, w_2) = (2/3, 1/3)$, the 2nd target hit and miss of the instance ⓪ are given by $(h_0^2, m_0^2) = (①, \boxed{5})$. Since $d(x_0, x_1|w) < d(x_0, x_5|w)$ under this weighted metric, the instance ⓪ is correctly classified in 3NN classification.

## 3 Algorithm

The problem (2) can be formulated as a convex quadratic program (QP) if TNs are regarded as fixed. Based on this fact, our feature weighting algorithm solves a sequence of such QPs, while TNs are properly updated to be always consistent.

### 3.1 Active Set QP Formulation

First, we study the problem (2) under the condition that TNs remain unchanged. Let us define the following sets of indices:

**Definition 4** *Given a weight vector $w$ and the consistent TNs $\{(h_i^\kappa, m_i^\lambda)\}_{i \in \mathbb{N}_n}$, define the following sets of index pairs for '$*$' being '$<$', '$=$', and '$>$':*

$$\mathcal{H}^{[*]} := \{(i, h) \in \mathbb{N}_n \times \mathbb{H}_i \mid d(x_i, x_h|w) * d(x_i, x_{h_i^\kappa}|w)\},$$

$$\mathcal{M}^{[*]} := \{(i, m) \in \mathbb{N}_n \times \mathbb{M}_i \mid d(x_i, x_m|w) * d(x_i, x_{m_i^\lambda}|w)\}.$$

*They are collectively denoted by $(\mathcal{H}, \mathcal{M})$, where $\mathcal{H} := \{\mathcal{H}^{[<]}, \mathcal{H}^{[=]}, \mathcal{H}^{[>]}\}$ and $\mathcal{M} := \{\mathcal{M}^{[<]}, \mathcal{M}^{[=]}, \mathcal{M}^{[>]}\}$. Furthermore, for each $i \in \mathbb{N}_n$, we define $\mathcal{H}_i^{[*]} := \{h|(i, h) \in \mathcal{H}^{[*]}\}$ and $\mathcal{M}_i^{[*]} := \{m|(i, m) \in \mathcal{M}^{[*]}\}$.*

Under the condition that $\{(h_i^\kappa, m_i^\lambda)\}_{i \in \mathbb{N}_n}$ remain to be TN-weight consistent, the problem (2) is written as

$$\min_{w \in \mathbb{R}^\ell, \xi \in \mathbb{R}^n, \eta \in \mathbb{R}^n} \quad \theta n^{-1} \sum_{i \in \mathbb{N}_n} (\xi_i - \eta_i) + \frac{1}{2}||w - \bar{w}||_2^2 \tag{3a}$$

$$\text{s.t.} \quad \mathbf{1}^\top w = 1, \ w \geq \mathbf{0}, \tag{3b}$$

$$d(x_i, x_h|w) \leq \xi_i, (i, h) \in \mathcal{H}^{[<]}, \ d(x_i, x_m|w) \leq \eta_i, (i, m) \in \mathcal{M}^{[<]}, \tag{3c}$$

$$d(x_i, x_h|w) = \xi_i, (i, h) \in \mathcal{H}^{[=]}, \ d(x_i, x_m|w) = \eta_i, (i, m) \in \mathcal{M}^{[=]}, \tag{3d}$$

$$d(x_i, x_h|w) \geq \xi_i, (i, h) \in \mathcal{H}^{[>]}, \ d(x_i, x_m|w) \geq \eta_i, (i, m) \in \mathcal{M}^{[>]}. \tag{3e}$$

In the above, we introduced slack variables $\xi_i$ and $\eta_i$ for $i \in \mathbb{N}_n$ which represent the weighted distances to the target hit and miss, respectively. In (3), TN-weight consistency is represented by a set of linear constraints (3c)–(3e)[3].

Our algorithm handles TN change as a change in the index sets $(\mathcal{H}, \mathcal{M})$, and a sequence of convex QPs in the form of (3) are (partially) solved every time the index sets $(\mathcal{H}, \mathcal{M})$ are updated. We implement this approach by using an *active set QP algorithm* (see Chapter 16 in [15]). Briefly, the active set QP algorithm repeats the following two steps: (step1) Estimate the optimal active set[4], and (step2) Solve an equality-constrained QP by regarding the constraints in the current active set as equality constraints and all the other non-active constraints are temporarily disregarded. An advantage of introducing the active set QP algorithm is that TN change can be naturally handled as active set change. Specifically, a change of target hits is interpreted as an exchange of the members between $\mathcal{H}^{[<]}$ and $\mathcal{H}^{[=]}$ or between $\mathcal{H}^{[>]}$ and $\mathcal{H}^{[=]}$, while a change of target misses is interpreted as an exchange of the members between $\mathcal{M}^{[<]}$ and $\mathcal{M}^{[=]}$ or between $\mathcal{M}^{[>]}$ and $\mathcal{M}^{[=]}$.

## 3.2 Sequential QP-based Feature Weighting Algorithm

Here, we present our feature weighting algorithm. We first formulate the equality-constrained QP (EQP) of (3). Then we present how to update the EQP by changing the active sets.

In order to formulate the EQP of (3), we introduce another pair of index sets $\mathcal{Z} := \{j|w_j = 0\}$ and $\mathcal{P} := \{j|w_j > 0\}$. Suppose that we currently have a solution $(w, \xi, \eta)$ and the active set $(\mathcal{H}^{[=]}, \mathcal{M}^{[=]}, \mathcal{Z})$. We first check whether the solution minimizes the loss function (3a) in the subspace defined by the active set. If not, we compute a step $(\Delta w, \Delta \xi, \Delta \eta)$ by solving an EQP:

$$\min_{\Delta w, \Delta \xi, \Delta \eta} \theta n^{-1} \sum_{i \in \mathbb{N}_n} ((\xi_i + \Delta \xi_i) - (\eta_i + \Delta \eta_i)) + \frac{1}{2} ||(w + \Delta w) - \bar{w}||_2^2$$

$$\text{s.t.} \quad \mathbf{1}^\top (w + \Delta w) = 1, \; w_j + \Delta w_j = 0, \; j \in \mathcal{Z}, \tag{4}$$

$$\varepsilon_{i,h}^\top (w + \Delta w) = \xi_i + \Delta \xi_i, \; (i, h) \in \mathcal{H}^{[=]}, \; \varepsilon_{i,m}^\top (w + \Delta w) = \eta_i + \Delta \eta_i, \; (i, m) \in \mathcal{M}^{[=]}.$$

The solution of the EQP (4) can be analytically obtained by solving a small linear system (see Supplement A).

Next, we decide how far we can move the solution along this direction. We set $w \leftarrow w + \tau \Delta w$, $\xi \leftarrow \xi + \tau \Delta \xi$, $\eta \leftarrow \eta + \tau \Delta \eta$, where $\tau \in [0, 1]$ is the step-length determined by the following lemma.

**Lemma 5** *The maximum step length that satisfies feasibility and TN-weight consistency is given by*

$$\tau := \min\Big( 1, \min_{j \in \mathcal{P}, \Delta w_j < 0} \frac{-w_j}{\Delta w_j},$$

$$\min_{(i,h) \in \mathcal{H}^{[<]}, \varepsilon_{i,h}^\top \Delta w > \Delta \xi_i} \frac{-(\varepsilon_{i,h}^\top w - \xi_i)}{\varepsilon_{i,h}^\top \Delta w - \Delta \xi_i}, \min_{(i,h) \in \mathcal{H}^{[>]}, \varepsilon_{i,h}^\top \Delta w < \Delta \xi_i} \frac{-(\varepsilon_{i,h}^\top w - \xi_i)}{\varepsilon_{i,h}^\top \Delta w - \Delta \xi_i}, \tag{5}$$

$$\min_{(i,m) \in \mathcal{M}^{[<]}, \varepsilon_{i,m}^\top \Delta w > \Delta \eta_i} \frac{-(\varepsilon_{i,m}^\top w - \eta_i)}{\varepsilon_{i,m}^\top \Delta w - \Delta \eta_i}, \min_{(i,m) \in \mathcal{M}^{[>]}, \varepsilon_{i,m}^\top \Delta w < \Delta \eta_i} \frac{-(\varepsilon_{i,m}^\top w - \eta_i)}{\varepsilon_{i,m}^\top \Delta w - \Delta \eta_i} \Big).$$

The proof of the lemma is presented in Supplement B.

If $\tau < 1$, the constraint for which the minimum in (5) is achieved (called the *blocking constraint*) is added to the active set. For example, if $(i, h) \in \mathcal{H}^{[>]}$ achieved the minimum in (5), $(i, h)$ is moved from $\mathcal{H}^{[>]}$ to $\mathcal{H}^{[=]}$. We repeat this by adding constraints to the active set until we reach the solution $(w, \xi, \eta)$ that minimizes the objective function over the current active set.

Next, we need to consider whether the objective function of (2) can be further decreased by removing constraints in the active set. Our algorithm and the standard active set QP algorithm are different in this operation: in our algorithm, an active constraint is allowed to be inactive only when the $\kappa$-th target hit remains to be a member of $\mathcal{H}^{[=]}$ and the $\lambda$-th target miss remains to be a member of $\mathcal{M}^{[=]}$.

Let us introduce the Lagrange multipliers $\alpha \in \mathbb{R}^{|\mathcal{Z}|}$, $\beta \in \mathbb{R}^{|\mathcal{H}^{[=]}|}$, and $\gamma \in \mathbb{R}^{|\mathcal{M}^{[=]}|}$ for the 2nd, the 3rd, and the 4th constraint in (4), respectively (see Supplement A for details). Then the following lemma tells us which active constraint should be removed.

**Lemma 6** *The objective function in (2) can be further decreased while satisfying feasibility and TN-weight consistency by removing one of the constraints in the active set with the following rules[5]:*

- *If $\alpha_j > 0$ for $j \in \mathcal{Z}$, then move $\{j\}$ to $\mathcal{P}$;*
- *If $\beta_{(i,h)} < 0$, $|\mathcal{H}_i^{[<]}| \leq \kappa - 2$ and $|\mathcal{H}_i^{[=]}| \geq 2$ for $(i, h) \in \mathcal{H}^{[=]}$, then move $(i, h)$ to $\mathcal{H}^{[<]}$;*
- *If $\beta_{(i,h)} > 0$, $|\mathcal{H}_i^{[>]}| < |\mathbb{H}_i| - \kappa$ and $|\mathcal{H}_i^{[=]}| \geq 2$ for $(i, h) \in \mathcal{H}^{[=]}$, then move $(i, h)$ to $\mathcal{H}^{[>]}$;*
- *If $\gamma_{(i,m)} < 0$, $|\mathcal{M}_i^{[<]}| \leq \lambda - 2$ and $|\mathcal{M}_i^{[=]}| \geq 2$ for $(i, m) \in \mathcal{M}^{[=]}$, then move $(i, m)$ to $\mathcal{M}^{[<]}$;*
- *If $\gamma_{(i,m)} > 0$, $|\mathcal{M}_i^{[>]}| < |\mathbb{M}_i| - \lambda$ and $|\mathcal{M}_i^{[=]}| \geq 2$ for $(i, m) \in \mathcal{M}^{[=]}$, then move $(i, m)$ to $\mathcal{M}^{[>]}$.*

The proof of the lemma is presented in Supplement C.

The proposed feature weighting algorithm, which we call *Sequential QP-based Feature Weighting (SQP-FW) algorithm*, is summarized in Algorithm 1. The proposed SQP-FW algorithm possesses

---

**Algorithm 1** Sequential QP-based Feature Weighting (SQP-FW) Algorithm

---

**Inputs**: The training instances $\{(x_i, y_i)\}_{i \in \mathbb{N}_n}$, the neighborhood parameters $(\kappa, \lambda)$, regularization parameter $\theta$, and initial weight vector $\bar{w}$;
Initialize $w \leftarrow \bar{w}$, $(\xi, \eta)$ and $(\mathcal{H}, \mathcal{M}, \mathcal{Z}, \mathcal{P})$;
**for** $t = 1, 2, \ldots$ **do**
    Solve (4) to find $(\Delta w, \Delta \xi, \Delta \eta)$;
   **if** $(\Delta w, \Delta \xi, \Delta \eta) = \mathbf{0}$ **then**
        Compute Lagrange multipliers $\alpha$, $\beta$, and $\gamma$;
        **if** none of the active constraints satisfies the rules in Lemma 6 **then**
            **stop** with solution $w^* = w$;
        **else**
            Update $(\mathcal{H}, \mathcal{M}, \mathcal{Z}, \mathcal{P})$ according to the rule in Lemma 6;
   **else**
        Compute the step size $\tau$ as in Lemma 5;
        **if** there are blocking constraints **then**
            Update $(\mathcal{H}, \mathcal{M}, \mathcal{Z}, \mathcal{P})$ by adding one of the blocking constraints in Lemma 5;
**Outputs**: A local optimal vector of feature weights $w^*$.

---

the following useful properties.

**Optimality conditions**: We can characterize a local optimal solution of the non-smooth and non-convex problem (2) in the following theorem (its proof is presented in Supplement D):

**Theorem 7 (Optimality condition)** *Consider a weight vector $w$ satisfying $\mathbf{1}^\top w = 1$ and $w \geq \mathbf{0}$, the consistent TNs $\{(h_i^\kappa, m_i^\lambda)\}_{i \in \mathbb{N}_n}$, and the index sets $(\mathcal{H}, \mathcal{M}, \mathcal{Z}, \mathcal{P})$. Then, $w$ is a local minimum solution of the problem (2) if and only if the EQP (4) has the solution $(\Delta w, \Delta \xi, \Delta \eta) = \mathbf{0}$ and there are no active constraints that satisfy the rules in Lemma 6.*

This theorem is practically useful because it guarantees that the solution cannot be improved in its neighborhood even if some of the current TNs are replaced with others. Without such an optimality condition, we must check all possible combinations of TN change from the current solution in a trial and error manner. The above theorem allows us to avoid such time-consuming procedure.

**Finite termination property**: It can be shown that the SQP-FW algorithm converges to a local minimum solution characterized by Theorem 7 in a finite number of iterations based on the similar argument as that in pages 477–478 in [15]. See Supplement E for details.

**Computational complexity**: When computing the solutions $(\Delta w, \Delta \xi, \Delta \eta)$ and the Lagrange multipliers $(\alpha, \beta, \gamma)$ by solving the EQP (4), the main computational cost is only several matrix-vector multiplications involving $n \times |\mathcal{P}|$ and $n \times |\mathcal{Z}|$ matrices, which is linear with respect to $n$ (see Supplement A for details). On the other hand, if the minimum step length $\tau$ is computed naively by Lemma 5, it takes $\mathcal{O}(n^2 |\mathcal{P}|)$ computations, which could be a bottleneck of the algorithm. However, this bottleneck can be eased by introducing a working set approach: only a fixed number of constraints in the working set are evaluated at each step, while the working set is updated, say, every 100 steps. In our implementation, we introduced such working sets to $\mathcal{H}^{[>]}$ and $\mathcal{M}^{[>]}$. For each $i \in \mathbb{N}_n$, these working sets contain, say, only top 100 nearest instances. This strategy is based on a natural idea that those outside of the top 100 nearest instances would not become TNs in the next 100 steps. Such a working set strategy allows us to reduce the computational complexity to $\mathcal{O}(n|\mathcal{P}|)$ for computing the the minimum step length $\tau$, which is linear with respect to $n$.

**Regularization path tracking**: The SQP-FW algorithm can be naturally combined with *regularization path tracking* algorithm for computing a path of the solutions that satisfy the optimality condition in Theorem 7 for a range of regularization parameter $\theta$. Due to the space limitation, we only describe the outline here (see Supplement F for details). The algorithm starts from a local optimal solution for a fixed regularization parameter $\theta$. Then, the algorithm continues finding the optimal solutions when $\theta$ is slightly increased. It can be shown that the local optimal solution of (2)

is a *piecewise-linear* function of $\theta$ as long as the TNs remain unchanged. If $\theta$ is further increased, we encounter a point at which TNs must be updated. Such TN changes can be easily detected and handled because the TN-weight consistency conditions are represented by a set of linear constraints (see (3c)–(3e)), and we already have explicit rules (Lemmas 5 and 6) for updating the constraints. The regularization path tracking algorithm provides an efficient and insightful approach for model selection.

## 4 Experiments

In this section, we investigate the experimental performance of the proposed algorithm[6].

### 4.1 Comparison Using UCI Data Sets

First, we compare the proposed SQP-FW algorithm with existing feature weighting algorithms, which handle the TN-change problem in different ways.

• `Relief` [7, 8]: The Relief algorithm is an online feature weighting algorithm. The goal of Relief is to maximize the average $(1,1)$-neighbor margin over instances. The TNs $\{(h_i^1, m_i^1)\}_{i \in \mathbb{N}_n}$ are determined by the initial Euclidean metric and fixed all through the training process.

• `Simba` [9]: Simba is also an online algorithm aiming to maximize the average $(1,1)$-neighbor margin. The key difference from Relief is that TNs $\{(h_i^1, m_i^1)\}_{i \in \mathbb{N}_n}$ are updated in each step using the current feature-space metric. The TN-change problem is alleviated in Simba by this reassignment.

• `MulRel`: To mitigate the TN-weight inconsistency in Relief, we repeat the Relief procedure using the TNs defined by the learned weights in the previous loop (see also [5]).

• `NCA-D` [4]: Neighborhood component analysis with diagonal metric, which is essentially the same as *I-Relief* [10, 11]. Instead of discretely assigning TNs, the probability of an instance being TNs is considered. Using these stochastic neighbors, the average margin is formulated as a continuous (non-convex) function of the weights, by which the TN change problem is mitigated.

We compared the NN classification performance of these 4 algorithms and the SQP-FW algorithm on 10 UCI benchmark data sets summarized in Table 1. In each data set, we randomly divided the entire data set into the training, validation, and test sets with equal sizes. The number of neighbors $k \in \{1, 3, 5\}$ was selected based on the classification performance on the validation set.

In the SQP-FW algorithm, the neighborhood parameter $(\kappa, \lambda)$ and the regularization parameter $\theta$ were also determined to maximize the classification accuracy on the validation set. The neighborhood parameter $(\kappa, \lambda)$ were chosen from $\{(1,1), (2,2), (3,3)\}$, while $\theta$ was chosen from 100 evenly allocated candidates in log-scale between $10^{-3}$ and $10^0$. The working set strategy was used when $n > 1000$ with the working set size 100 and the working set update frequency 100.

All the 4 existing algorithms do not have explicit hyper-parameters. However, since these algorithms also have the risk of overfitting, we removed features with small weights, following the recommendation in [7, 11]. We implemented this heuristic for all the 4 existing algorithms by optimizing the percentage of eliminating features (chosen from $\{0\%, 1\%, 2\%, \dots, 99\%\}$) based on the classification performance on the validation set. Since Simba and NCA are formulated as non-convex optimization problems and solutions may be trapped in local minima, we ran these two algorithms from five randomly selected starting points and the solution with the smallest training error was adopted. The number of iterations in Relief (and the inner-loop iteration of MulRel as well) and Simba was set to 1000, and the outer-loop iteration of MulRel was set to 100.

The experiments were repeated 10 times with random data splitting, and the average performance was reported. To see the statistical significance of the difference, paired-sample $t$-test was conducted. All the features were standardized to have zero mean and unit variance. Table 1 summarizes the results, showing that the SQP-FW algorithm compares favorably with other methods.

Table 1: Average misclassification rate of $k$NN classifier on 10 UCI benchmark data sets.

| Abbreviated Data Name | S.S. | $\ell$ | N.C. | SQP-FW | Relief | Simba | MulRel | NCA-D |
|---|---|---|---|---|---|---|---|---|
| *Bre. Can. Dia.* | 569 | 30 | 2 | ***0.040** | **0.047** | **0.046** | **0.056** | 0.058 |
| *Con. Ben.* | 208 | 60 | 2 | ***0.221** | **0.227** | **0.230** | 0.294 | **0.276** |
| *Ima. Seg.* | 2310 | 18 | 7 | **0.052** | ***0.049** | 0.061 | 0.065 | **0.049** |
| *Ionosphere* | 351 | 33 | 2 | 0.122 | 0.162 | **0.115** | 0.138 | ***0.097** |
| *Pag. Blo. Cla.* | 5473 | 10 | 5 | **0.046** | 0.048 | ***0.044** | 0.053 | **0.044** |
| *Parkinson* | 195 | 22 | 2 | ***0.102** | **0.117** | 0.123 | **0.109** | 0.128 |
| *Pen. Rec. Han. Dig.* | 10992 | 16 | 10 | ***0.011** | **0.012** | 0.012 | 0.020 | 0.029 |
| *Spambase* | 4601 | 57 | 2 | ***0.104** | **0.108** | **0.110** | 0.117 | 0.112 |
| *Wav. Dat. Gen. ver1* | 5000 | 21 | 3 | ***0.184** | 0.202 | 0.217 | 0.227 | 0.195 |
| *Win. Qua.* | 6497 | 11 | 7 | ***0.463** | 0.499 | 0.471 | 0.494 | 0.495 |

'S.S.' and 'N.C.' stand for sample size and the number of classes, respectively. Asterisk '*' indicates the best among 5 algorithms, while boldface means no statistical difference from the best ($p$-value $\geq 0.05$).

Table 2: Results on Microarray Data Experiments

| Microarray Data Name | S.S. | $\ell$ | N.C. | Standard 1NN | Weighted 1NN with SQP-FW | |
|---|---|---|---|---|---|---|
| | | | | Error | Error | Med. #(genes) |
| Colon Cancer [16] | 62 | 2000 | 2 | $0.180 \pm 0.059$ | $0.140 \pm 0.065$ | 20 |
| Kidney Cancer [17] | 74 | 4224 | 3 | $0.075 \pm 0.043$ | $0.050 \pm 0.038$ | 10 |
| Leukemia [18] | 72 | 7129 | 2 | $0.108 \pm 0.022$ | $0.088 \pm 0.036$ | 14 |
| Prostate Cancer [19] | 102 | 12600 | 2 | $0.230 \pm 0.048$ | $0.194 \pm 0.052$ | 24 |

respectively. 'Error' represents the misclassification error rate of 1NN classifier, while 'Med. #(genes)' indicates the median number of genes selected by SQP-FW algorithm over 10 runs.

## 4.2 Application to Feature Selection Problem in High-Dimensional Microarray Data

In order to illustrate feature selection performance, we applied the SQP-FW algorithm to microarray study, in which simple classification algorithms are often preferred because the number of features (genes) $\ell$ is usually much larger than the number of instances (patients) $n$. Since biologists are interested in identifying a set of genes that governs the difference among different biological phenotypes (such as cancer subtypes), selecting a subset of genes that yields good NN classification performance would be practically valuable.

For each of the four microarray data sets in Table 2, we divided the entire set into the training and test sets with size ratio 2:1 [2]. We compared the test set classification performance between the plain 1NN classifier (without feature weighting) and the weighted 1NN classifier with the weights determined by the SQP-FW algorithm. In the latter, the neighborhood parameters were fixed to $\kappa = \lambda = 1$ and $\theta$ was determined by 10-fold cross validation within the training set. We repeated the data splitting 10 times and the average performance was reported.

Table 2 summarizes the results. The median numbers of selected genes (features with nonzero weights) by the SQP-FW algorithm are also reported in the table. Although the improvements of the classification performances were not statistically significant (we could not expect much improvement by feature weighting because the misclassification rates of the plain 1NN classifier are already very low), the number of genes used for NN classification can be greatly reduced. The results illustrate the potential advantage of feature selection using the SQP-FW algorithm.

## 5 Discussion and Conclusion

TN change is a fundamental problem in feature extraction and selection for NN classifiers. Our contribution in this paper was to present a feature weighting algorithm that can systematically handle TN changes and guarantee the TN-weight consistency. An important future direction is to generalize our TN-weight consistent feature weighting scheme to feature extraction (i.e., metric learning).

## Acknowledgment

IT was supported by MEXT KAKENHI 21200001 and 23700165, and MS was supported by MEXT KAKENHI 23120004.

## Footnotes

[1] The terminologies *target hit* and *miss* were first used in [7], in which only the 1st target hit and miss were considered. We extend them to the $\kappa$-th target hit and $\lambda$-th target miss for general $\kappa$ and $\lambda$. The terminology *target neighbors (TNs)* was first used in [5].

[2] The notion of the nearest neighbor margin was first introduced in [9], where only the case of $\kappa = \lambda = 1$ was considered. We use an extended definition with general $\kappa$ and $\lambda$.

[3] Note that the constraints for $(\mathcal{H}^{[<]}, \mathcal{H}^{[=]}, \mathcal{H}^{[>]})$ in (3c)–(3e) restrict that $h$ must remain to be the target hit of $i$ for all $(i, h) \in \mathcal{H}^{[=]}$ because those closer than the target hit must remain to be closer and those more distant than the target hit must remain to be more distant. Similarly, the constraints for $(\mathcal{M}^{[<]}, \mathcal{M}^{[=]}, \mathcal{M}^{[>]})$ in (3c)–(3e) restrict that $m$ must remain to be the target miss of $i$ for all $(i, m) \in \mathcal{M}^{[=]}$.

[4] A constraint satisfied with equality is called *active* and the set of active constraints is called *active set*.

[5]If multiple active constraints are selected by these rules, the one with the largest absolute Lagrange multiplier is removed from the active set.

[6]See also Supplement G for an illustration of the behavior of the proposed algorithm using an artificial dataset.

# References

[1] A. S. Das, M. Datar, A. Garg, and S. Rajaram. Google news personalization: Scalable online collaborative filtering. In *Proceedings of the 16th International Conference on World Wide Web*, pages 271–280. ACM, 2007.

[2] S. Dudoit, J. Fridlyand, and T. P. Speed. Comparison of discrimination methods for the classification of tumors using gene expression data. *Journal of the American Statistical Association*, 97(457):77–87, 2002.

[3] E. P. Xing, A. Y. Ng, M. I. Jordan, and S. Russell. Distance metric learning with application to clustering with side-information. In S. Thrun S. Becker and K. Obermayer, editors, *Advances in Neural Information Processing Systems 15*, pages 505–512. MIT Press, Cambridge MA, 2003.

[4] J. Goldberger, S. Roweis, G. Hinton, and R. Salakhutdinov. Neighbourhood components analysis. In L. K. Saul, Y. Weiss, and L. Bottou, editors, *Advances in Neural Information Processing Systems 17*, pages 513–520. MIT Press, Cambridge, MA, 2005.

[5] K. Weinberger, J. Blitzer, and L. Saul. Distance metric learning for large margin nearest neighbor classification. In Y. Weiss, B. Schölkopf, and J. Platt, editors, *Advances in Neural Information Processing Systems 18*, pages 1473–1480. MIT Press, Cambridge, MA, 2006.

[6] J. Davis, B. Kulis, P. Jain, S. Sra, and I. Dhillon. Information-theoretic metric learning. In *Proceedings of the 24th International Conference on Machine Learning*, pages 209–216, 2007.

[7] K. Kira and L. Rendell. A practical approach to feature selection. In *Proceedings of the 9-th International Conference on Machine Learning*, pages 249–256, 1992.

[8] I. Kononenko. Estimating attributes: analysis and extensions of relief. In *Proceedings of European Conference on Machine Learning*, pages 171–182, 1994.

[9] R. Gilad-Bachrach, A. Navot, and N. Tishby. Margin based feature selection - theory and algorithms. In *Proceedings of the 21st International Conference on Machine Learning*, pages 43–50, 2004.

[10] Y. Sun and J. Li. Iterative relief for feature weighting. In *Proceedings of the 23-rd International Conference on Machhine Learning*, pages 913–920, 2006.

[11] Y. Sun, S. Todorovic, and S. Goodison. Local learning based feature selection for high dimensional data analysis. *IEEE Transactions on Pattern Analysis and Machine Intelligence*, 32(9):1610–1626, 2010.

[12] K. Wagsta, C. Cardie, S. Rogers, and S. Schroedl. Constrained k-means clustering with background knowledge. In *Proceedings of the Eighteenth International Conference on Machine Learning*, pages 577–584, 2001.

[13] M. Sugiyama. Dimensionality reduction of multimodal labeled data by local fisher discriminant analysis. *Journal of Machine Learning Research*, 8:1027–1061, 2007.

[14] T. Hastie, S. Rosset, R. Tibshirani, and J. Zhu. The entire regularization path for the support vector machine. *Journal of Machine Learning Research*, 5:1391–1415, 2004.

[15] J. Nocedal and S. J. Wright. *Numerical optimization*. Springer, 1999.

[16] U. Alon, N. Barkia, D.A. Notterman, and K. Gish et al. Broad patterns of gene expression revealed by clustering analysis of tumor and normal colon tissues probed by oligonucleotide arrays. *Proc. Natl. Acad. Sci. USA*, 96:6745–6750, 1999.

[17] H. Sueltmann, A. Heydenbreck, W. Huber, and R. Kuner et al. Gene expression in kidney cancer is associated with novel tumor subtypes, cytogenetic abnormalities and metastasis formation. 8:1027–1061, 2007.

[18] T. R. Golub, D. K. Slonim, P. Tamayo, and C. Huard et al. Molecular classification of cancer: class discovery and class prediction by gene expression monitoring. *Science*, 286:531–537, 1999.

[19] D. Singh, P. G. Febbo, K. Ross, and D. G. Jackson et al. Gene expression correlates of clinical prostate cancer behavior. *Cancer Cell*, 1:203–209, 2002.

